# Semi-Markov Conditional Random Fields for Information Extraction

**Sunita Sarawagi**
Indian Institute of Technology
Bombay, India
sunita@iitb.ac.in

**William W. Cohen**
Center for Automated Learning & Discovery
Carnegie Mellon University
wcohen@cs.cmu.edu

## Abstract

We describe semi-Markov conditional random fields (semi-CRFs), a conditionally trained version of semi-Markov chains. Intuitively, a semi-CRF on an input sequence $\mathbf{x}$ outputs a "segmentation" of $\mathbf{x}$, in which labels are assigned to segments (*i.e.*, subsequences) of $\mathbf{x}$ rather than to individual elements $x_i$ of $\mathbf{x}$. Importantly, features for semi-CRFs can measure properties of segments, and transitions within a segment can be non-Markovian. In spite of this additional power, exact learning and inference algorithms for semi-CRFs are polynomial-time—often only a small constant factor slower than conventional CRFs. In experiments on five named entity recognition problems, semi-CRFs generally outperform conventional CRFs.

## 1 Introduction

Conditional random fields (CRFs) are a recently-introduced formalism [12] for representing a conditional model $\Pr(\mathbf{y}|\mathbf{x})$, where both $\mathbf{x}$ and $\mathbf{y}$ have non-trivial structure (often sequential). Here we introduce a generalization of sequential CRFs called *semi-Markov conditional random fields* (or semi-CRFs). Recall that *semi-Markov chain models* extend hidden Markov models (HMMs) by allowing each state $s_i$ to persist for a non-unit length of time $d_i$. After this time has elapsed, the system will transition to a new state $s'$, which depends only on $s_i$; however, during the "segment" of time between $i$ and $i + d_i$, the behavior of the system may be non-Markovian. Generative semi-Markov models are fairly common in certain applications of statistics [8, 9], and are also used in reinforcement learning to model hierarchical Markov decision processes [19].

Semi-CRFs are a conditionally trained version of semi-Markov chains. In this paper, we present inference and learning methods for semi-CRFs. We also argue that segments often have a clear intuitive meaning, and hence semi-CRFs are more natural than conventional CRFs. We focus here on named entity recognition (NER), in which a segment corresponds to an extracted entity; however, similar arguments might be made for several other tasks, such as gene-finding [11] or NP-chunking [16].

In NER, a semi-Markov formulation allows one to easily construct entity-level features (such as "entity length" and "similarity to other known entities") which cannot be easily encoded in CRFs. Experiments on five different NER problems show that semi-CRFs often outperform conventional CRFs.

## 2 CRFs and Semi-CRFs

### 2.1 Definitions

A CRF models $\Pr(\mathbf{y}|\mathbf{x})$ using a Markov random field, with nodes corresponding to elements of the structured object $\mathbf{y}$, and potential functions that are conditional on (features of) $\mathbf{x}$. Learning is performed by setting parameters to maximize the likelihood of a set of $(\mathbf{x}, \mathbf{y})$ pairs given as training data. One common use of CRFs is for sequential learning problems like NP chunking [16], POS tagging [12], and NER [15]. For these problems the Markov field is a chain, and $\mathbf{y}$ is a linear sequence of labels from a fixed set $\mathcal{Y}$. For instance, in the NER application, $\mathbf{x}$ might be a sequence of words, and $\mathbf{y}$ might be a sequence in $\{I, O\}^{|\mathbf{x}|}$, where $y_i = I$ indicates "word $x_i$ is inside a name" and $y_i = O$ indicates the opposite.

Assume a vector $\mathbf{f}$ of *local feature functions* $\mathbf{f} = \langle f^1, \ldots, f^K \rangle$, each of which maps a pair $(\mathbf{x}, \mathbf{y})$ and an index $i$ to a measurement $f^k(i, \mathbf{x}, \mathbf{y}) \in R$. Let $\mathbf{f}(i, \mathbf{x}, \mathbf{y})$ be the vector of these measurements, and let $\mathbf{F}(\mathbf{x}, \mathbf{y}) = \sum_i^{|\mathbf{x}|} \mathbf{f}(i, \mathbf{x}, \mathbf{y})$. For example, in NER, the components of $\mathbf{f}$ might include the measurement $f^{13}(i, \mathbf{x}, \mathbf{y}) = [\![x_i \text{ is capitalized}]\!] \cdot [\![y_i = I]\!]$, where the indicator function $[\![c]\!] = 1$ if $c$ if true and zero otherwise; this implies that $F^{13}(\mathbf{x}, \mathbf{y})$ would be the number of capitalized words $x_i$ paired with the label $I$. Following previous work [12, 16] we will define a conditional random field (CRF) to be an estimator of the form

$$\Pr(\mathbf{y}|\mathbf{x}, \mathbf{W}) = \frac{1}{Z(\mathbf{x})} e^{\mathbf{W} \cdot \mathbf{F}(\mathbf{x}, \mathbf{y})} \tag{1}$$

where $\mathbf{W}$ is a weight vector over the components of $\mathbf{F}$, and $Z(\mathbf{x}) = \sum_{\mathbf{y}'} e^{\mathbf{W} \cdot \mathbf{F}(\mathbf{x}, \mathbf{y}')}$.

To extend this to the semi-Markov case, let $\mathbf{s} = \langle s_1, \ldots, s_p \rangle$ denote a *segmentation* of $\mathbf{x}$, where *segment* $s_j = \langle t_j, u_j, y_j \rangle$ consists of a *start position* $t_j$, an *end position* $u_j$, and a *label* $y_j \in Y$. Conceptually, a segment means that the tag $y_j$ is given to all $x_i$'s between $i = t_j$ and $i = u_j$, inclusive. We assume segments have positive length, and completely cover the sequence $1 \ldots |\mathbf{x}|$ without overlapping: that is, that $t_j$ and $u_j$ always satisfy $t_1 = 1, u_p = |\mathbf{x}|, 1 \le t_j \le u_j \le |\mathbf{x}|$, and $t_{j+1} = u_j + 1$. For NER, a valid segmentation of the sentence "I went skiing with Fernando Pereira in British Columbia" might be $\mathbf{s} = \langle (1, 1, O), (2, 2, O), (3, 3, O), (4, 4, O), (5, 6, I), (7, 7, O), (8, 9, I) \rangle$, corresponding to the label sequence $\mathbf{y} = \langle O, O, O, O, I, I, O, I, I \rangle$.

We now assume a vector $\mathbf{g}$ of *segment feature functions* $\mathbf{g} = \langle g^1, \ldots, g^K \rangle$, each of which maps a triple $(j, \mathbf{x}, \mathbf{s})$ to a measurement $g^k(j, \mathbf{x}, \mathbf{s}) \in R$, and define $\mathbf{G}(\mathbf{x}, \mathbf{s}) = \sum_j^{|\mathbf{s}|} \mathbf{g}(j, \mathbf{x}, \mathbf{s})$. We also make a restriction on the features, analogous to the usual Markovian assumption made in CRFs, and assume that every component $g^k$ of $\mathbf{g}$ is a function only of $\mathbf{x}$, $s_j$, and the label $y_{j-1}$ associated with the preceding segment $s_{j-1}$. In other words, we assume that every $g^k(j, \mathbf{x}, \mathbf{s})$ can be rewritten as

$$g^k(j, \mathbf{x}, \mathbf{s}) = g'^k(y_j, y_{j-1}, \mathbf{x}, t_j, u_j) \tag{2}$$

for an appropriately defined $g'^k$. In the rest of the paper, we will drop the $g'$ notation and use $g$ for both versions of the segment-level feature functions. A *semi-CRF* is then an estimator of the form

$$\Pr(\mathbf{s}|\mathbf{x}, \mathbf{W}) = \frac{1}{Z(\mathbf{x})} e^{\mathbf{W} \cdot \mathbf{G}(\mathbf{x}, \mathbf{s})} \tag{3}$$

where again $\mathbf{W}$ is a weight vector for $\mathbf{G}$ and $Z(\mathbf{x}) = \sum_{\mathbf{s}'} e^{\mathbf{W} \cdot \mathbf{G}(\mathbf{x}, \mathbf{s}')}$.

### 2.2 An efficient inference algorithm

The *inference problem* for a semi-CRF is defined as follows: given $\mathbf{W}$ and $\mathbf{x}$, find the best segmentation, $argmax_{\mathbf{s}} \Pr(\mathbf{s}|\mathbf{x}, \mathbf{W})$, where $\Pr(\mathbf{s}|\mathbf{x}, \mathbf{W})$ is defined by Equation 3. An

efficient inference algorithm is suggested by Equation 2, which implies that

$$argmax_{\mathbf{s}} \Pr(\mathbf{s}|\mathbf{x}, \mathbf{W}) = argmax_{\mathbf{s}} \mathbf{W} \cdot \mathbf{G}(\mathbf{x}, \mathbf{s}) = argmax_{\mathbf{s}} \mathbf{W} \cdot \sum_j \mathbf{g}(y_j, y_{j-1}, \mathbf{x}, t_j, u_j)$$

Let $L$ be an upper bound on segment length. Let $\mathbf{s}_{i:y}$ denote the set of all partial segmentations starting from 1 (the first index of the sequence) to $i$, such that the last segment has the label $y$ and ending position $i$. Let $V_{\mathbf{x},\mathbf{g},W}(i, y)$ denote the largest value of $\mathbf{W} \cdot \mathbf{G}(\mathbf{x}, \mathbf{s}')$ for any $\mathbf{s}' \in \mathbf{s}_{i:y}$. Omitting the subscripts, the following recursive calculation implements a semi-Markov analog of the usual Viterbi algorithm:

$$V(i, y) = \begin{cases} \max_{y', d=1...L} \ V(i - d, y') + \mathbf{W} \cdot \mathbf{g}(y, y', \mathbf{x}, i - d + 1, i) & \text{if } i > 0 \\ 0 & \text{if } i = 0 \\ -\infty & \text{if } i < 0 \end{cases} \quad (4)$$

The best segmentation then corresponds to the path traced by $\max_y V(|\mathbf{x}|, y)$.

### 2.3 Semi-Markov CRFs *vs* order-$L$ CRFs

Since conventional CRFs need not maximize over possible segment lengths $d$, inference for semi-CRFs is more expensive. However, Equation 4 shows that the additional cost is only linear in $L$. For NER, a reasonable value of $L$ might be four or five.[1] Since in the worst case $L \leq |\mathbf{x}|$, the semi-Markov Viterbi algorithm is always polynomial, even when $L$ is unbounded.

For fixed $L$, it can be shown that semi-CRFs are no more expressive than order-$L$ CRFs. For order-$L$ CRFs, however the additional computational cost is exponential in $L$. The difference is that semi-CRFs only consider sequences in which the *same* label is assigned to all $L$ positions, rather than all $|\mathcal{Y}|^L$ length-$L$ sequences. This is a useful restriction, as it leads to faster inference.

Semi-CRFs are also a natural restriction, as it is often convenient to express features in terms of segments. As an example, let $d_j$ denote the length of a segment, and let $\mu$ be the average length of all segments with label $I$. Now consider the segment feature $g^{k_1}(j, \mathbf{x}, \mathbf{s}) = (d_j - \mu)^2 \cdot [\![y_j = I]\!]$. After training, the contribution of this feature toward $\Pr(\mathbf{s}|\mathbf{x})$ associated with a length-$d$ entity will be proportional to $e^{w_k \cdot (d-\mu)^2}$—i.e., it allows the learner to model a Gaussian distribution of entity lengths.

An exponential model for lengths could be implemented with the feature $g^{k_2}(j, \mathbf{x}, \mathbf{y}) = d_j \cdot [\![y_j = I]\!]$. In contrast to the Gaussian-length feature above, $g^{k_2}$ is "equivalent to" a local feature function $f(i, \mathbf{x}, \mathbf{y}) = [\![y_i = I]\!]$, in the following sense: for every triple $\mathbf{x}, \mathbf{y}, \mathbf{s}$, where $\mathbf{y}$ is the tags for $\mathbf{s}$, $\sum_j g^{k_2}(j, \mathbf{x}, \mathbf{s}) = \sum_i f(i, \mathbf{s}, \mathbf{y})$. Thus a semi-CRF model based on the single feature $g^{k_2}$ could also be represented by a conventional CRF.

In general, a semi-CRF model can be factorized in terms of an equivalent order-1 CRF model if and only if the sum of the segment features can be rewritten as a sum of local features. Thus the degree to which semi-CRFs are non-Markovian depends on the feature set.

### 2.4 Learning algorithm

During training the goal is to maximize log-likelihood over a given training set $T = \{(\mathbf{x}_\ell, \mathbf{s}_\ell)\}_{\ell=1}^N$. Following the notation of Sha and Pereira [16], we express the log-likelihood over the training sequences as

$$L(\mathbf{W}) = \sum_\ell \log \Pr(\mathbf{s}_\ell|\mathbf{x}_\ell, \mathbf{W}) = \sum_\ell (\mathbf{W} \cdot \mathbf{G}(\mathbf{x}_\ell, \mathbf{s}_\ell) - \log Z_{\mathbf{W}}(\mathbf{x}_\ell)) \quad (5)$$

We wish to find a $\mathbf{W}$ that maximizes $L(\mathbf{W})$. Equation 5 is convex, and can thus be maximized by gradient ascent, or one of many related methods. (In our implementation we use a limited-memory quasi-Newton method [13, 14].) The gradient of $L(\mathbf{W})$ is the following:

$$\nabla L(\mathbf{W}) \;=\; \sum_{\ell} \mathbf{G}(\mathbf{x}_\ell, \mathbf{s}_\ell) - \frac{\sum_{\mathbf{s}'} \mathbf{G}(\mathbf{s}', \mathbf{x}_\ell) e^{\mathbf{W} \cdot \mathbf{G}(\mathbf{x}_\ell, \mathbf{s}')}}{Z_{\mathbf{W}}(\mathbf{x}_\ell)} \tag{6}$$

$$\;=\; \sum_{\ell} \mathbf{G}(\mathbf{x}_\ell, \mathbf{s}_\ell) - E_{\mathrm{Pr}(\mathbf{s}'|\mathbf{W})} \mathbf{G}(\mathbf{x}_\ell, \mathbf{s}') \tag{7}$$

The first set of terms are easy to compute. However, to compute the the normalizer, $Z_{\mathbf{W}}(\mathbf{x}_\ell)$, and the expected value of the features under the current weight vector, $E_{\mathrm{Pr}(\mathbf{s}'|\mathbf{W})} \mathbf{G}(\mathbf{x}_\ell, \mathbf{s}')$, we must use the Markov property of $\mathbf{G}$ to construct a dynamic programming algorithm, similar for forward-backward. We thus define $\alpha(i, y)$ as the value of $\sum_{\mathbf{s}' \in \mathbf{s}_{i:y}} e^{\mathbf{W} \cdot \mathbf{G}(\mathbf{s}', \mathbf{x})}$ where again $\mathbf{s}_{i:y}$ denotes all segmentations from 1 to $i$ ending at $i$ and labeled $y$. For $i > 0$, this can be expressed recursively as

$$\alpha(i, y) = \sum_{d=1}^{L} \sum_{y' \in \mathcal{Y}} \alpha(i - d, y') e^{\mathbf{W} \cdot \mathbf{g}(y, y', \mathbf{x}, i-d+1, i)}$$

with the base cases defined as $\alpha(0, y) = 1$ and $\alpha(i, y) = 0$ for $i < 0$. The value of $Z_{\mathbf{W}}(\mathbf{x})$ can then be written as $Z_{\mathbf{W}}(\mathbf{x}) = \sum_y \alpha(|\mathbf{x}|, y)$.

A similar approach can be used to compute the expectation $\sum_{\mathbf{s}'} \mathbf{G}(\mathbf{x}_\ell, \mathbf{s}') e^{\mathbf{W} \cdot \mathbf{G}(\mathbf{x}_\ell, \mathbf{s}')}$. For the $k$-th component of $\mathbf{G}$, let $\eta^k(i, y)$ be the value of the sum $\sum_{\mathbf{s}' \in \mathbf{s}_{i:y}} G^k(\mathbf{s}', \mathbf{x}_\ell) e^{\mathbf{W} \cdot \mathbf{G}(\mathbf{x}_\ell, \mathbf{s}')}$, restricted to the part of the segmentation ending at position $i$. The following recursion[2] can then be used to compute $\eta^k(i, y)$:

$\eta^k(i, y) =$
$\quad \sum_{d=1}^{L} \sum_{y' \in \mathcal{Y}} (\eta^k(i-d, y') + \alpha(i-d, y') g^k(y, y', \mathbf{x}, i-d+1, i)) e^{\mathbf{W} \cdot \mathbf{g}(y, y', \mathbf{x}, i-d+1, i)}$

Finally we let $E_{\mathrm{Pr}(\mathbf{s}'|\mathbf{W})} G^k(\mathbf{s}', \mathbf{x}) = \frac{1}{Z_{\mathbf{W}}(\mathbf{x})} \sum_y \eta^k(|\mathbf{x}|, y)$.

## 3  Experiments with NER data

### 3.1  Baseline algorithms and datasets

In our experiments, we trained semi-CRFs to mark entity segments with the label $I$, and put non-entity words into unit-length segments with label $O$. We compared this with two versions of CRFs. The first version, which we call CRF/1, labels words inside and outside entities with $I$ and $O$, respectively. The second version, called CRF/4, replaces the $I$ tag with four tags $B$, $E$, $C$, and $U$, which depend on where the word appears in an entity [2].

We compared the algorithms on five NER problems, associated with three different corpora. The *Address* corpus contains 4,226 words, and consists of 395 home addresses of students in a major university in India [1]. We considered extraction of city names and state names from this corpus. The *Jobs* corpus contains 73,330 words, and consists of 300 computer-related job postings [4]. We considered extraction of company names and job titles. The 18,121-word *Email* corpus contains 216 email messages taken from the CSPACE email corpus [10], which is mail associated with a 14-week, 277-person management game. Here we considered extraction of person names.

### 3.2 Features

As features for CRF, we used indicators for specific words at location $i$, or locations within three words of $i$. Following previous NER work [7]), we also used indicators for capitalization/letter patterns (such as "Aa+" for a capitalized word, or "D" for a single-digit number).

As features for semi-CRFs, we used the same set of word-level features, as well their logical extensions to segments. Specifically, we used indicators for the phrase inside a segment and the capitalization pattern inside a segment, as well as indicators for words and capitalization patterns in 3-word windows before and after the segment. We also used indicators for each segment length ($d = 1, \ldots, L$), and combined all word-level features with indicators for the beginning and end of a segment.

To exploit more of the power of semi-CRFs, we also implemented a number of dictionary-derived features, each of which was based on different dictionary $D$ and similarity function $sim$. Letting $\mathbf{x}_{s_j}$ denote the subsequence $\langle x_{t_j} \ldots x_{u_j} \rangle$, a dictionary feature is defined as $g^{D,sim}(j, \mathbf{x}, \mathbf{s}) = argmax_{u \in D} sim(\mathbf{x}_{s_j}, u)$—i.e., the distance from the word sequence $\mathbf{x}_{s_j}$ to the closest element in $D$.

For each of the extraction problems, we assembled one *external dictionary* of strings, which were similar (but not identical) to the entity names in the documents. For instance, for city names in the *Address data*, we used a web page listing cities in India. Due to variations in the way entity names are written, rote matching these dictionaries to the data gives relatively low F1 values, ranging from 22% (for the job-title extraction task) to 57% (for the person-name task). We used three different similarity metrics (Jaccard, TFIDF, and JaroWinkler) which are known to work well for name-matching in data integration tasks [5]. All of the distance metrics are non-Markovian—i.e., the distance-based segment features cannot be decomposed into sums of local features. More detail on the distance metrics, feature sets, and datasets above can be found elsewhere [6].

We also extended the semi-CRF algorithm to construct, on the fly, an *internal segment dictionary* of segments labeled as entities in the training data. To make measurements on training data similar to those on test data, when finding the closest neighbor of $\mathbf{x}_{s_j}$ in the internal dictionary, we excluded all strings formed from $\mathbf{x}$, thus excluding matches of $\mathbf{x}_{s_j}$ to itself (or subsequences of itself). This feature could be viewed as a sort of nearest-neighbor classifier; in this interpretation the semi-CRF is performing a sort of bi-level stacking [21].

For completeness in the experiments, we also evaluated local versions of the dictionary features. Specifically, we constructed dictionary features of the form $f^{D,sim}(i, \mathbf{x}, \mathbf{y}) = argmax_{u \in D} sim(x_i, u)$, where $D$ is either the external dictionary used above, or an *internal word dictionary* formed from all words contained in entities. As before, words in $\mathbf{x}$ were excluded in finding near neighbors to $x_i$.

### 3.3 Results and Discussion

We evaluated F1-measure performance[3] of CRF/1, CRF/4, and semi-CRFs, with and without internal and external dictionaries. A detailed tabulation of the results are shown in Table 1, and Figure 1 shows F1 values plotted against training set size for a subset of three of the tasks, and four of the learning methods. In each experiment performance was averaged over seven runs, and evaluation was performed on a hold-out set of 30% of the documents. In the table the learners are trained with 10% of the available data—as the curves show, performance differences are often smaller with more training data. Gaussian priors were used for all algorithms, and for semi-CRFs, a fixed value of $L$ was chosen for each dataset based on observed entity lengths. This ranged between 4 and 6 for the different datasets.

In the baseline configuration in which no dictionary features are used, semi-CRFs perform

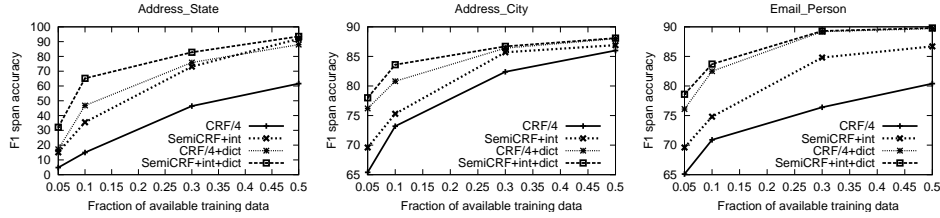

Figure 1: F1 as a function of training set size. Algorithms marked with "+dict" include external dictionary features, and algorithms marked with "+int" include internal dictionary features. We do not use internal dictionary features for CRF/4 since they lead to reduced accuracy.

| | baseline | +internal dict | | +external dict | | +both dictionaries | | |
|---|---|---|---|---|---|---|---|---|
| | F1 | F1 | Δbase | F1 | Δbase | F1 | Δbase | Δextern |
| **CRF/1** | | | | | | | | |
| state | 20.8 | **44.5** | 113.9 | **69.2** | 232.7 | 55.2 | 165.4 | -67.3 |
| title | 28.5 | 3.8 | -86.7 | 38.6 | 35.4 | 19.9 | -30.2 | -65.6 |
| person | 67.6 | 48.0 | -29.0 | 81.4 | 20.4 | 64.7 | -4.3 | -24.7 |
| city | 70.3 | 60.0 | -14.7 | 80.4 | 14.4 | 69.8 | -0.7 | -15.1 |
| company | 51.4 | 16.5 | -67.9 | 55.3 | 7.6 | 15.6 | -69.6 | -77.2 |
| **CRF/4** | | | | | | | | |
| state | 15.0 | 25.4 | 69.3 | 46.8 | 212.0 | 43.1 | 187.3 | -24.7 |
| title | 23.7 | 7.9 | -66.7 | 36.4 | 53.6 | 14.6 | -38.4 | -92.0 |
| person | 70.9 | 64.5 | -9.0 | 82.5 | 16.4 | 74.8 | 5.5 | -10.9 |
| city | 73.2 | 70.6 | -3.6 | 80.8 | 10.4 | 76.3 | 4.2 | -6.1 |
| company | 54.8 | 20.6 | -62.4 | **61.2** | 11.7 | 25.1 | -54.2 | -65.9 |
| **semi-CRF** | | | | | | | | |
| state | **25.6** | 35.5 | 38.7 | 62.7 | 144.9 | **65.2** | 154.7 | 9.8 |
| title | **33.8** | **37.5** | 10.9 | **41.1** | 21.5 | **40.2** | 18.9 | -2.5 |
| person | **72.2** | **74.8** | 3.6 | **82.8** | 14.7 | **83.7** | 15.9 | 1.2 |
| city | **75.9** | **75.3** | -0.8 | **84.0** | 10.7 | **83.6** | 10.1 | -0.5 |
| company | **60.2** | **59.7** | -0.8 | 60.9 | 1.2 | **60.9** | 1.2 | 0.0 |

Table 1: Comparing various methods on five IE tasks, with and without dictionary features. The column Δbase is percentage change in F1 values relative to the baseline. The column Δextern is is change relative to using only external-dictionary features.

best on all five of the tasks. When internal dictionary features are used, the performance of semi-CRFs is often improved, and never degraded by more than 2.5%. However, the less-natural local version of these features often leads to substantial performance losses for CRF/1 and CRF/4. Semi-CRFs perform best on nine of the ten task variants for which internal dictionaries were used. The external-dictionary features are helpful to all the algorithms. Semi-CRFs performs best on three of five tasks in which only external dictionaries were used.

Overall, semi-CRF performs quite well. If we consider the tasks with and without external dictionary features as separate "conditions", then semi-CRFs using all available information[4] outperform both CRF variants on eight of ten "conditions".

We also compared semi-CRF to order-$L$ CRFs, with various values of $L$.[5] In Table 2 we show the result for $L = 1$, $L = 2$, and $L = 3$, compared to semi-CRF. For these tasks, the performance of CRF/4 and CRF/1 does not seem to improve much by simply increasing

| | CRF/1 | | | CRF/4 | | | semi-CRF |
|---|---|---|---|---|---|---|---|
| | $L = 1$ | $L = 2$ | $L = 3$ | $L = 1$ | $L = 2$ | $L = 3$ | |
| Address_State | 20.8 | 20.1 | 19.2 | 15.0 | 16.4 | 16.4 | **25.6** |
| Address_City | 70.3 | 71.0 | 71.2 | 73.2 | 73.9 | 73.7 | **75.9** |
| Email_persons | 67.6 | 63.7 | 66.7 | 70.9 | 70.7 | 70.4 | **72.2** |

Table 2: F1 values for different order CRFs

order.

## 4   Related work

Semi-CRFs are similar to nested HMMs [1], which can also be trained discrimini-tively [17]. The primary difference is that the "inner model" for semi-CRFs is of short, uniformly-labeled segments with non-Markovian properties, while nested HMMs allow longer, diversely-labeled, Markovian "segments".

Discriminative learning methods can be used for conditional random fields with architectures more complex than chains (*e.g.*, [20, 18]), and one of these methods has also been applied to NER [3]. Further, by creating a random variable for each possible segment, one can learn models strictly more expressive than the semi-Markov models described here. However, for these methods, inference is not tractable, and hence approximations must be made in training and classification. An interesting question for future research is whether the tractible extension to CRF inference considered here can can be used to improve inference methods for more expressive models.

In recent prior work [6], we investigated semi-Markov learning methods for NER based on a voted perceptron training algorithm [7]. The voted perceptron has some advantages in ease of implementation, and efficiency. (In particular, the voted perceptron algorithm is more stable numerically, as it does not need to compute a partition function. ) However, semi-CRFs perform somewhat better, on average, than our perceptron-based learning algorithm. Probabilistically-grounded approaches like CRFs also are preferable to margin-based approaches like the voted perceptron in certain settings, *e.g.*, when it is necessary to estimate confidences in a classification.

## 5   Concluding Remarks

Semi-CRFs are a tractible extension of CRFs that offer much of the power of higher-order models without the associated computational cost. A major advantage of semi-CRFs is that they allow features which measure properties of segments, rather than individual elements. For applications like NER and gene-finding [11], these features can be quite natural.

### Appendix

An implementation of semi-CRFs is available at http://crf.sourceforge.net, and a NER package using this package is available on http://minorthird.sourceforge.net.

## Footnotes

[1]Assuming that non-entity words are placed in unit-length segments, as we do below.

[2]As in the forward-backward algorithm for chain CRFs [16], space requirements here can be reduced from $ML|\mathcal{Y}|$ to $M|\mathcal{Y}|$, where $M$ is the length of the sequence, by pre-computing an appropriate set of $\beta$ values.

[3]F1 is defined as 2*precision*recall/(precision+recall.)

[4]*I.e.*, the both-dictionary version when external dictionaries are available, and the internal-dictionary only version otherwise.

[5]Order-$L$ CRFs were implemented by replacing the label set $\mathcal{Y}$ with $\mathcal{Y}^L$. We limited experiments to $L \leq 3$ for computational reasons.

# References

[1] V. R. Borkar, K. Deshmukh, and S. Sarawagi. Automatic text segmentation for extracting structured records. In *Proc. ACM SIGMOD International Conf. on Management of Data*, Santa Barabara,USA, 2001.

[2] A. Borthwick, J. Sterling, E. Agichtein, and R. Grishman. Exploiting diverse knowledge sources via maximum entropy in named entity recognition. In *Sixth Workshop on Very Large Corpora New Brunswick, New Jersey. Association for Computational Linguistics.*, 1998.

[3] R. Bunescu and R. J. Mooney. Relational markov networks for collective information extraction. In *Proceedings of the ICML-2004 Workshop on Statistical Relational Learning (SRL-2004)*, Banff, Canada, July 2004.

[4] M. E. Califf and R. J. Mooney. Bottom-up relational learning of pattern matching rules for information extraction. *Journal of Machine Learning Research*, 4:177–210, 2003.

[5] W. W. Cohen, P. Ravikumar, and S. E. Fienberg. A comparison of string distance metrics for name-matching tasks. In *Proceedings of the IJCAI-2003 Workshop on Information Integration on the Web (IIWeb-03)*, 2003.

[6] W. W. Cohen and S. Sarawagi. Exploiting dictionaries in named entity extraction: Combining semi-markov extraction processes and data integration methods. In *Proceedings of the Tenth ACM SIGKDD International Conference on Knowledge Discovery and Data Mining*, 2004.

[7] M. Collins. Discriminative training methods for hidden Markov models: Theory and experiments with perceptron algorithms. In *Empirical Methods in Natural Language Processing (EMNLP)*, 2002.

[8] X. Ge. *Segmental Semi-Markov Models and Applications to Sequence Analysis*. PhD thesis, University of California, Irvine, December 2002.

[9] J. Janssen and N. Limnios. *Semi-Markov Models and Applications*. Kluwer Academic, 1999.

[10] R. E. Kraut, S. R. Fussell, F. J. Lerch, and J. A. Espinosa. Coordination in teams: evidence from a simulated management game. To appear in the Journal of Organizational Behavior, 2005.

[11] A. Krogh. Gene finding: putting the parts together. In M. J. Bishop, editor, *Guide to Human Genome Computing*, pages 261–274. Academic Press, 2nd edition, 1998.

[12] J. Lafferty, A. McCallum, and F. Pereira. Conditional random fields: Probabilistic models for segmenting and labeling sequence data. In *Proceedings of the International Conference on Machine Learning (ICML-2001)*, Williams, MA, 2001.

[13] D. C. Liu and J. Nocedal. On the limited memory BFGS method for large-scale optimization. *Mathematic Programming*, 45:503–528, 1989.

[14] R. Malouf. A comparison of algorithms for maximum entropy parameter estimation. In *Proceedings of The Sixth Conference on Natural Language Learning (CoNLL-2002)*, pages 49–55, 2002.

[15] A. McCallum and W. Li. Early results for named entity recognition with conditional random fields, feature induction and web-enhanced lexicons. In *Proceedings of The Seventh Conference on Natural Language Learning (CoNLL-2003)*, Edmonton, Canada, 2003.

[16] F. Sha and F. Pereira. Shallow parsing with conditional random fields. In *Proceedings of HLT-NAACL*, 2003.

[17] M. Skounakis, M. Craven, and S. Ray. Hierarchical hidden Markov models for information extraction. In *Proceedings of the 18th International Joint Conference on Artificial Intelligence, Acapulco, Mexico. Morgan Kaufmann.*, 2003.

[18] C. Sutton, K. Rohanimanesh, and A. McCallum. Dynamic conditional random fields: Factorized probabilistic models for labeling and segmenting sequence data. In *ICML*, 2004.

[19] R. Sutton, D. Precup, and S. Singh. Between MDPs and semi-MDPs: A framework for temporal abstraction in reinforcement learning. *Artificial Intelligence*, 112:181–211, 1999.

[20] B. Taskar, P. Abbeel, and D. Koller. Discriminative probabilistic models for relational data. In *Proceedings of Eighteenth Conference on Uncertainty in Artificial Intelligence (UAI02)*, Edmonton, Canada, 2002.

[21] D. H. Wolpert. Stacked generalization. *Neural Networks*, 5:241–259, 1992.
